# Computational structure of coordinate transformations: A generalization study

**Zoubin Ghahramani**
zoubin@psyche.mit.edu

**Daniel M. Wolpert**
wolpert@psyche.mit.edu

**Michael I. Jordan**
jordan@psyche.mit.edu

Department of Brain & Cognitive Sciences
Massachusetts Institute of Technology
Cambridge, MA 02139

## Abstract

One of the fundamental properties that both neural networks and the central nervous system share is the ability to learn and generalize from examples. While this property has been studied extensively in the neural network literature it has not been thoroughly explored in human perceptual and motor learning. We have chosen a coordinate transformation system—the visuomotor map which transforms visual coordinates into motor coordinates—to study the generalization effects of learning new input–output pairs. Using a paradigm of computer controlled altered visual feedback, we have studied the generalization of the visuomotor map subsequent to both local and context-dependent remappings. A local remapping of one or two input-output pairs induced a significant global, yet decaying, change in the visuomotor map, suggesting a representation for the map composed of units with large functional receptive fields. Our study of context-dependent remappings indicated that a single point in visual space can be mapped to two different finger locations depending on a context variable—the starting point of the movement. Furthermore, as the context is varied there is a gradual shift between the two remappings, consistent with two visuomotor modules being learned and gated smoothly with the context.

## 1   Introduction

The human central nervous system (CNS) receives sensory inputs from a multitude of modalities, each tuned to extract different forms of information from the

environment. These sensory signals are initially represented in disparate coordinate systems, for example visual information is represented retinotopically whereas auditory information is represented tonotopically. The ability to transform information between coordinate systems is necessary for both perception and action. When we reach to a visually perceived object in space, for example, the location of the object in visual coordinates must be converted into a representation appropriate for movement, such as the configuration of the arm required to reach the object. The computational structure of this coordinate transformation, known as the visuomotor map, is the focus of this paper.

By examining the change in visuomotor coordination under prismatically induced displacement and rotation, Helmholtz (1867/1925) and Stratton (1897a,1897b) pioneered the systematic study of the representation and plasticity of the visuomotor map. Their studies demonstrate both the fine balance between the visual and motor coordinate systems, which is disrupted by such perturbations, and the ability of the visuomotor map to adapt to the displacements induced by the prisms. Subsequently, many studies have further demonstrated the remarkable plasticity of the map in response to a wide variety of alterations in the relationship between the visual and motor system (for reviews see Howard, 1982 and Welch, 1986)—the single prerequisite for adaptation seems to be that the remapping be stable (Welch, 1986). Much less is known, however, about the topological properties of this map.

A coordinate transformation such as the visuomotor map can be regarded as a function relating one set of variables (inputs) to another (outputs). For the visuomotor map the inputs are visual coordinates of a desired target and the outputs are the corresponding motor coordinates representing the arm's configuration (e.g. joint angles). The problem of learning a sensorimotor remapping can then be regarded as a function approximation problem. Using the theory of function approximation one can make explicit the correspondence between the representation used and the patterns of generalization that will emerge. Function approximators can predict patterns of generalization ranging from local (look-up tables), through intermediate (CMACs, Albus, 1975; and radial basis functions, Moody and Darken, 1989 ) to global (parametric models).

In this paper we examine the representational structure of the visuomotor map through the study of its spatial and contextual generalization properties. In the spatial generalization study we address the question of how pointing changes over the reaching workspace after exposure to a highly localized remapping. Previous work on spatial generalization, in a study restricted to one dimension, has led to the conclusion that the visuomotor map is constrained to generalize linearly (Bedford, 1989). We test this conclusion by mapping out the pattern of generalization induced by one and two remapped points in two dimensions.

In the contextual generalization study we examine the question of whether a single point in visual space can be mapped into two different finger locations depending on the context of a movement—the start point. If this context-dependent remapping occurs, the question arises as to how the mapping will generalize as the context is varied. Studies of contextual remapping have previously shown that variables such as eye position (Kohler, 1950; Hay and Pick, 1966; Shelhamer et al., 1991), the feel of prisms (Kravitz, 1972; Welch, 1971) or an auditory tone (Kravitz and Yaffe, 1972), can induce context-dependent aftereffects. The question of how these context-

dependent maps generalize—which has not been previously explored—reflects on the possible representation of multiple visuomotor maps and their mixing with a context variable.

## 2  Spatial Generalization

To examine the spatial generalization of the visuomotor map we measured the change in pointing behavior subsequent to one- and two-point remappings. In order to measure pointing behavior and to confine subjects to learn limited input-output pairs we used a virtual visual feedback setup consisting of a digitizing tablet to record the finger position on-line and a projection/mirror system to generate a cursor spot image representing the finger position (Figure 1a). By controlling the presence of the cursor spot and its relationship to the finger position, we could both restrict visual feedback of finger position to localized regions of space and introduce perturbations of this feedback.

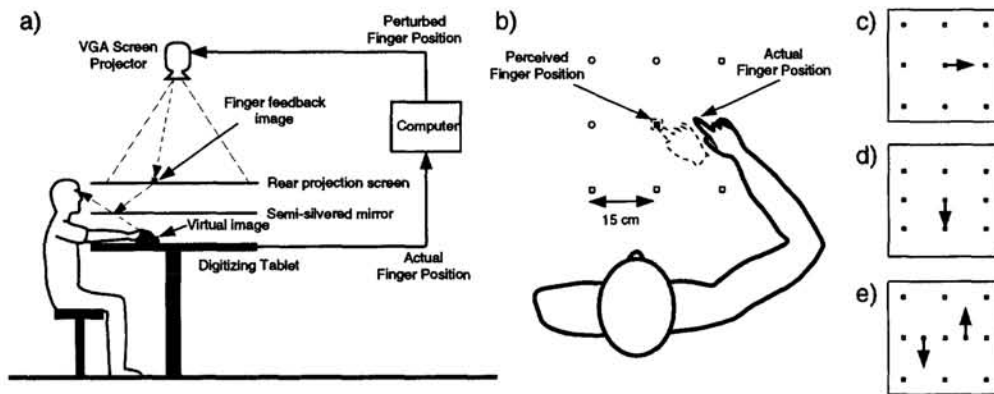

Figure 1.  a) Experimental setup.  The subjects view the reflected image of the rear projection screen by looking down at the mirror.  By matching the screen-mirror distance to the mirror-tablet distance all projected images appeared to be in the plane of the finger (when viewed in the mirror) independent of head position. b) The position of the grid of targets relative to the subject.  Also shown, for the $x$-shift condition, is the perceived and actual finger position when pointing to the central training target.  The visually perceived finger position is indicated by a cursor spot which is displaced from the actual finger position.  c) A schematic showing the perturbation for the $x$-shift group.  To see the cursor spot on the central target the subjects had to place their finger at the position indicated by the tip of the arrow. d) & e) Schematics similar to c) showing the perturbation for the $y$-shift and two point groups, respectively.

In the tradition of adaptation studies (e.g. Welch, 1986), each experimental session consisted of three phases: pre-exposure, exposure, and post-exposure.  During the pre- and post-exposure phases, designed to assess the visuomotor map, the subject pointed repeatedly, without visual feedback of his finger position, to a grid of targets over the workspace.  As no visual input of finger location was given, no learning of the visuomotor map could occur.  During the exposure phase subjects pointed repeatedly to one or two visual target locations, at which we introduced a discrep-

ancy between the actual and visually displayed finger location. No visual feedback of finger position was given except when within 0.5 cm of the target, thereby confining any learning to the chosen input-output pairs. Three local perturbations of the visuomotor map were examined: a 10 cm rightward displacement ($x$-shift group, Figure 1c), 10 cm displacement towards the body ($y$-shift group, Figure 1d), and a displacement at two points, one 10 cm away from, and one 10 cm towards the body (two point group, Figure 1e). For example, for the $x$-shift displacement the subject had to place his finger 10 cm to the right of the target to visually perceive his finger as being on target (Figure 1b). Separate control subjects, in which the relationship between the actual and visually displayed finger position was left unaltered, were run for both the one- and two-point displacements, resulting in a total of 5 groups with 8 subjects each.

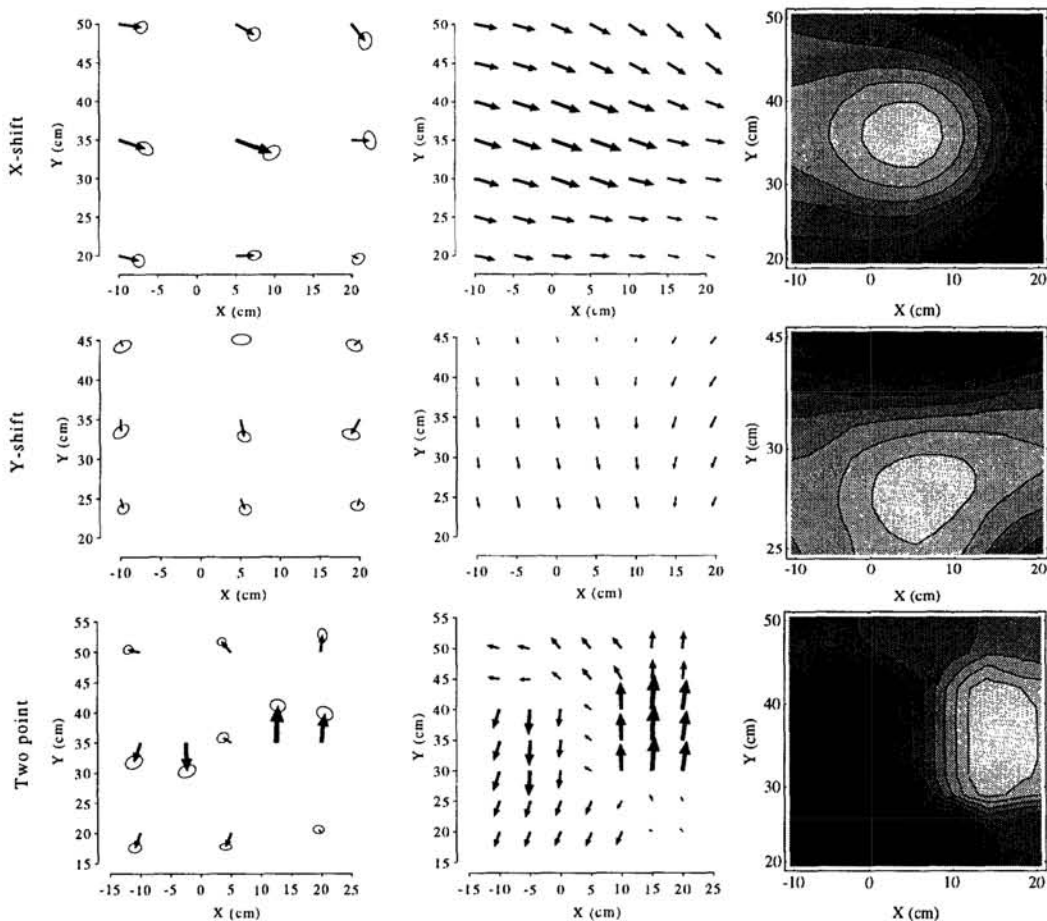

Figure 2. Results of the spatial generalization study. The first column shows the mean change in pointing, along with 95% confidence ellipses, for the $x$-shift, $y$-shift and two point groups. The second column displays a vector field of changes obtained from the data by Gaussian kernel smoothing. The third column plots the proportion adaptation in the direction of the perturbation. Note that whereas for the $x$- and $y$-shift groups the lighter shading corresponds to greater adaptation, for the two point group lighter shades correspond to adaptation in the positive $y$ direction and darker shades in the negative $y$ direction.

The patterns of spatial generalization subsequent to exposure to the three local remappings are shown in Figure 2. All three perturbation groups displayed both significant adaptation at the trained points, and significant, through decremented, generalization of this learning to other targets. As expected, the control groups (not shown) did not show any significant changes. The extent of spatial generalization, best depicted by the shaded contour plots in Figure 2, shows a pattern of generalization that decreases with distance away from the trained points. Rather than inducing a single global change in the map, such as a rotation or shear, the two point exposure appears to induce two opposite fields of decaying generalization at the intersection of which there is no change in the visuomotor map.

# 3 Contextual Generalization

The goal of this experiment was first to explore the possibility that multiple visuo-motor maps, or modules, could be learned, and if so, to determine how the overall system behaves as the context used in training each module is varied. To achieve this goal, we exposed subjects to context-dependent remappings in which a single visual target location was mapped to two different finger positions depending on the start point of the movement. Pointing to the target from seven different starting points (Figure 3) was assessed before and after an exposure phase. During this exposure phase subjects made repeated movements to the target from starting points 2 and 6 with a different perturbation of the visual feedback depending on the starting point. The form of these context-dependent remappings is shown in Figure 3. For example, for the open $x$-shift group (Figure 3c), the visual feedback of the finger was displaced to the right for movements from point 2 and to the left from point 6. Therefore the same visual target was mapped to two different finger positions depending on the context of the movement. To test learning of the remapping and generalization to other start points we examined the change in pointing, without visual feedback, to the target from the 7 start points.

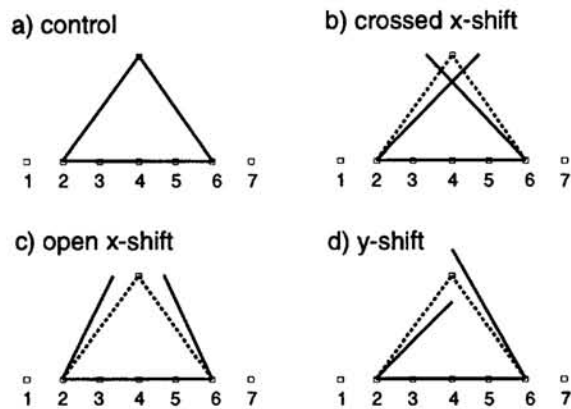

Figure 3. Schematic of the exposure phase in the contextual generalization experiment. Shown are the actual finger path (solid line), the visually displayed finger path (dotted line), the seven start points and the target used in the pre- and post-exposure phases. The perturbation introduced depended on whether the movement started form start point 2 or 6. Note that for the three perturbation groups, although the subjects saw a triangle being traced out, the finger took a different path.

The results are shown in Figure 4. Whereas the controls did not show any significant pattern of change, the three other groups showed adaptive, start point dependent, changes in the direction opposite to the perturbation. Thus, for example, the $x$-open group displayed a pattern of change in the leftward (negative $x$) direction for movements from the left start points and rightwards for movements from the right start points. Furthermore, as the start point was varied, the change in pointing varied gradually.

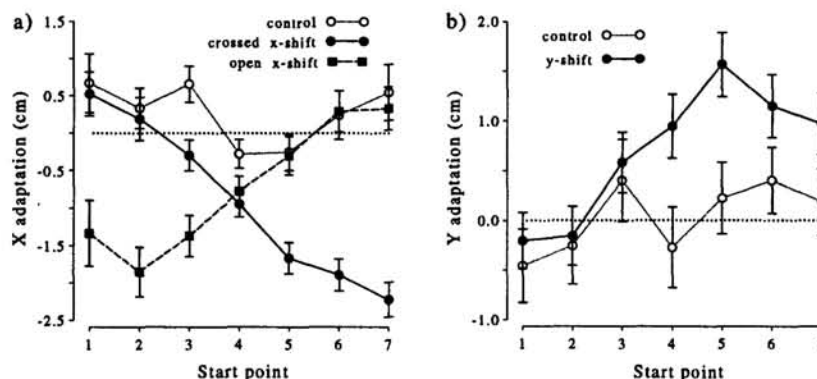

Figure 4. a) Adaptation in the $x$ direction plotted as a function of starting point for the control, crossed $x$-shift and open $x$-shift groups (mean and 1 s.e.). b) Adaptation in the $y$ direction for the control and $y$-shift groups.

## 4 Discussion

Clearly, from the perspective of function approximation theory, the problem of relearning the visuomotor mapping from exposure to one or two input-output pairs is ill-posed. The mapping learned, as measured by the pattern of generalization to novel inputs, therefore reflects intrinsic constraints on the internal representation used.

The results from the spatial generalization study suggest that the visuomotor coordinate transformation is internally represented with units with large but localized receptive fields. For example, a neural network model with Gaussian radial basis function units (Moody and Darken, 1989), which can be derived by assuming that the internal constraint in the visuomotor system is a smoothness constraint (Poggio and Girosi, 1989), predicts a pattern of generalization very similar the one experimentally observed (e.g. see Figure 5 for a simulation of the two point generalization experiment).[1] In contrast, previously proposed models for the representation of the visuomotor map based on global parametric representations in terms of felt direction of gaze and head position (e.g. Harris, 1965) or linear constraints (Bedford, 1989) do not predict the decaying patterns of Cartesian generalization found.

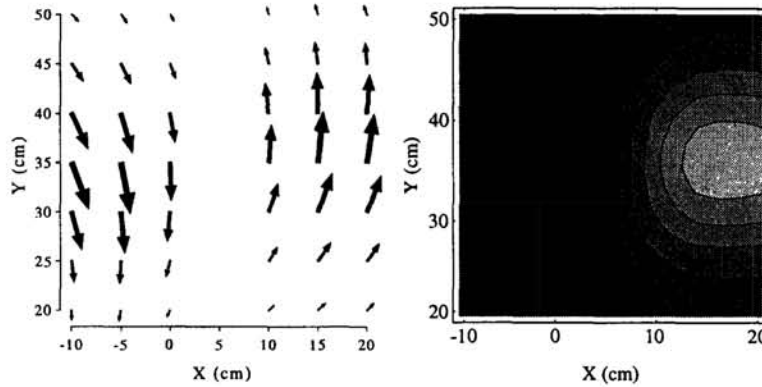

Figure 5. Simulation of the two point spatial generalization experiment using a radial basis function network with 64 units with 5 cm Gaussian receptive fields. The inputs to the network were the visual coordinates of the target and the outputs were the joint angles for a two-link planar arm to reach the target. The network was first trained to point accurately to the targets, and then, after exposure to the perturbation, its pattern of generalization was assessed.

The results from the second study suggest that multiple visuomotor maps can be learned and modulated by a context. A suggestive computational model for how such separate modules can be learned and combined is the mixture-of-experts neural network architecture (Jacobs et al., 1991). Interpreted in this framework, the gradual effect of varying the context seen in Figure 4 could reflect the output of a gating network which uses context to modulate between two visuomotor maps. However, our results do not rule out models in which a single visuomotor map is parametrized by starting location, such as one based on the coding of locations via movement vectors (Georgopoulos, 1990).

# 5 Conclusions

The goal of these studies has been to infer the internal constraints in the visuomotor system through the study of its patterns of generalization to local remappings. We have found that local perturbations of the visuomotor map produce global changes, suggesting a distributed representation with large receptive fields. Furthermore, context-dependent perturbations induce changes in pointing consistent with a model of visuomotor learning in which separate maps are learned and gated by the context. The approach taken in this paper provides a strong link between neural network theory and the study of learning in biological systems.

## Acknowledgements

This project was supported in part by a grant from the McDonnell-Pew Foundation, by a grant from ATR Human Information Processing Research Laboratories, by a grant from Siemens Corporation, and by grant N00014-94-1-0777 from the Office of Naval Research. Zoubin Ghahramani and Daniel M. Wolpert are McDonnell-Pew Fellows in Cognitive Neuroscience. Michael I. Jordan is a NSF Presidential Young Investigator.

## Footnotes

[1] See also Pouget & Sejnowski (this volume) who, based on a related analysis of neurophysiological data from parietal cortex, suggest that a basis function representation may be used in this visuomotor area.

# References

Albus, J. (1975). A new approach to manipulator control: The cerebellar model articulation controller (CMAC). *J. of Dynamic Systems, Measurement, and Control*, 97:220–227.

Bedford, F. (1989). Constraints on learning new mappings between perceptual dimensions. *J. of Experimental Psychology: Human Perception and Performance*, 15(2):232–248.

Georgopoulos, A. (1990). Neurophysiology of reaching. In Jeannerod, M., editor, *Attention and performance XIII*, pages 227–263. Lawrence Erlbaum, Hillsdale.

Harris, C. (1965). Perceptual adaptation to inverted, reversed, and displaced vision. *Psychological Review*, 72:419–444.

Hay, J. and Pick, H. (1966). Gaze-contingent prism adaptation: Optical and motor factors. *J. of Experimental Psychology*, 72:640–648.

Howard, I. (1982). *Human visual orientation*. Wiley, Chichester, England.

Jacobs, R., Jordan, M., Nowlan, S., and Hinton, G. (1991). Adaptive mixture of local experts. *Neural Computation*, 3:79–87.

Kohler, I. (1950). Development and alterations of the perceptual world: conditioned sensations. *Proceedings of the Austrian Academy of Sciences*, 227.

Kravitz, J. (1972). Conditioned adaptation to prismatic displacement. *Perception and Psychophysics*, 11:38–42.

Kravitz, J. and Yaffe, F. (1972). Conditioned adaptation to prismatic displacement with a tone as the conditional stimulus. *Perception and Psychophysics*, 12:305–308.

Moody, J. and Darken, C. (1989). Fast learning in networks of locally-tuned processing units. *Neural Computation*, 1(2):281–294.

Poggio, T. and Girosi, F. (1989). A theory of networks for approximation and learning. *AI Lab Memo 1140, MIT*.

Pouget, A. and Sejnowski, T. (1994). Spatial representation in the parietal cortex may use basis functions. In Tesauro, G., Touretzky, D., and Alspector, J., editors, *Advances in Neural Information Processing Systems 7*. Morgan Kaufmann.

Shelhamer, M., Robinson, D., and Tan, H. (1991). Context-specific gain switching in the human vestibuloocular reflex. In Cohen, B., Tomko, D., and Guedry, F., editors, *Annals Of The New York Academy Of Sciences*, volume 656, pages 889–891. New York Academy Of Sciences, New York.

Stratton, G. (1897a). Upright vision and the retinal image. *Psychological Review*, 4:182–187.

Stratton, G. (1897b). Vision without inversion of the retinal image. *Psychological Review*, 4:341–360, 463–481.

von Helmholtz, H. (1925). *Treatise on physiological optics (1867)*. Optical Society of America, Rochester, New York.

Welch, R. (1971). Discriminative conditioning of prism adaptation. *Perception and Psychophysics*, 10:90–92.

Welch, R. (1986). Adaptation to space perception. In Boff, K., Kaufman, L., and Thomas, J., editors, *Handbook of perception and performance*, volume 1, pages 24-1–24-45. Wiley–Interscience, New York.